# Non-linear CCA and PCA by Alignment of Local Models

**Jakob J. Verbeek[†], Sam T. Roweis[‡], and Nikos Vlassis[†]**
[†] Informatics Institute, University of Amsterdam
[‡] Department of Computer Science,University of Toronto

## Abstract

We propose a non-linear Canonical Correlation Analysis (CCA) method which works by coordinating or aligning mixtures of linear models. In the same way that CCA extends the idea of PCA, our work extends recent methods for non-linear dimensionality reduction to the case where multiple embeddings of the same underlying low dimensional coordinates are observed, each lying on a different high dimensional manifold. We also show that a special case of our method, when applied to only a single manifold, reduces to the Laplacian Eigenmaps algorithm. As with previous alignment schemes, once the mixture models have been estimated, all of the parameters of our model can be estimated in closed form without local optima in the learning. Experimental results illustrate the viability of the approach as a non-linear extension of CCA.

## 1   Introduction

In this paper, we are interested in data that lies on or close to a low dimensional manifold embedded, possibly non-linearly, in a Euclidean space of much higher dimension. Data of this kind is often generated when our observations are very high dimensional but the number of underlying degrees of freedom is small. A typical example are images of an object under different conditions (e.g. pose and lighting). A simpler example is given in Fig. 1, where we have data in $\mathbb{R}^3$ which lies on a two dimensional manifold. We want to recover the structure of the data manifold, so that we can 'unroll' the data manifold and work with the data expressed in the underlying 'latent coordinates', i.e. coordinates on the manifold. Learning low dimensional latent representations may be desirable for different reasons, such as compression for storage and communication, visualization of high dimensional data, or as preprocessing for further data analysis or prediction tasks.

Recent work on unsupervised nonlinear feature extraction has pursued several complementary directions. Various nonparametric spectral methods, such as Isomap[1], LLE[2], Kernel PCA[3] and Laplacian Eigenmaps[4] have been proposed which reduce the dimensionality of a fixed training set in a way that maximally preserve certain inter-point relationships, but these methods do not generally provide a functional mappings between the high and low dimensional spaces that are valid both on and off the training data. In this paper, we consider a method to integrate several local feature extractors into a single global representation, similar to the approaches of [5, 6, 7, 8]. These methods, as well as ours,

deliver after training a functional mapping which can be used to convert previously unseen high dimensional observations into their low dimensional global coordinates. Like most of the above algorithms, our method performs non-linear feature extraction by minimizing a convex objective function whose critical points can be characterized as eigenvectors of some matrix. These algorithms are generally simple and efficient; one needs only to construct a matrix based on local feature analysis of the training data and then computes its largest or smallest eigenvectors using standard numerical methods. In contrast, methods like generative topographic mapping[9] and self-organizing maps[10] are prone to local optima in the objective function.

Our method is based on the same intuitions as in earlier work: the idea is to learn a mixture of latent variable density models on the original training data so that each mixture component acts as a local feature extractor. For example, we may use a mixture of factor analyzers or a mixture of principal component analyzers (PCA). After this mixture has been learned, the local feature extractors are 'coordinated' by finding, for each model, a suitable linear mapping (and offset) from its latent variable space into a single 'global' low-dimensional coordinate system. The local feature extractors together with the coordinating linear maps provide a global non-linear map from the data space to the latent space and back. Learning the mixture is driven by a density signal – we want to place models near the training points, while the post-coordination is driven by the idea that when two different models place significant weight on the same point, they should agree on its mapping into the global space.

Our algorithm, developed in the following section, builds upon recent work of coordination methods. As in [6], we use a cross-entropy between a unimodal approximation and the true posterior over global coordinates to encourage agreement. However we do not attempt to simultaneously learn the mixture model and coordinate since this causes severe problems with local minima. Instead, as in [7, 8], we fix a specific mixture and then study the computations involved in coordinating its local representations. We extend the latter works as CCA extends PCA: rather than finding a projection of one set of points, we find projections for two sets of corresponding points $\{\mathbf{x}_n\}$ and $\{\mathbf{y}_n\}$ ($\mathbf{x}_n$ corresponding to $\mathbf{y}_n$) into a single latent space that project corresponding points in the two point sets as nearby as possible.

In this setting we begin by showing, in Section 3, how Laplacian Eigenmaps[4] are a special case of the algorithms presented here when they are applied to only a single manifold. We go on, in Section 4, to extend our algorithm to a setting in which multiple different observation spaces are available, each one related to the same underlying global space but through different nonlinear embeddings. This naturally gives rise to a nonlinear version of weighted Canonical Correlation Analysis (CCA). We present results of several experiments in the same section and we conclude the paper with a general discussion in Section 5.

## 2   Non-linear PCA by aligning local feature extractors

Consider a given data set $\mathbf{X} = \{\mathbf{x}_1, \dots, \mathbf{x}_N\}$ and a collection of $k$ local feature extractors, $\mathbf{f}_s(\mathbf{x})$ is a vector containing the, zero or more, features produced by model $s$. Each feature extractor also provides an "activity signal", $a_s(\mathbf{x})$ representing its confidence in modeling the point. We convert these activities into posterior responsibilities using a simple softmax: $p(s|\mathbf{x}) = \exp(a_s(\mathbf{x}))/\sum_r \exp(a_r(\mathbf{x}))$. If the experts are actually components of a mixture, then setting the activities to the logarithm of the posteriors under the mixture will recover exactly the same posteriors above.

Next, we consider the relationship between the given representation of the data and the representation of the data in a global latent space, which we would like to find. Throughout, we will use $\mathbf{g}$ to denote latent 'Global' coordinates for data. For the unobserved latent coordinate $\mathbf{g}$ corresponding to a data point $\mathbf{x}_n$ and conditioned on $s$, we assume the density:

$$p(\mathbf{g}|\mathbf{x}_n, s) = \mathcal{N}(\mathbf{g}; \boldsymbol{\kappa}_s + \mathbf{A}_s \mathbf{f}_s(\mathbf{x}_n), \sigma^2 \mathbf{I}) = \mathcal{N}(\mathbf{g}; \mathbf{g}_{ns}, \sigma^2 \mathbf{I}), \qquad (1)$$

where $\mathcal{N}(\mathbf{g}; \boldsymbol{\mu}, \boldsymbol{\Sigma})$ is a Gaussian distribution on $\mathbf{g}$ with mean $\boldsymbol{\mu}$ and covariance $\boldsymbol{\Sigma}$. The mean, $\mathbf{g}_{ns}$, of $p(\mathbf{g}|\mathbf{x}_n, s)$ is the sum of the component offset $\boldsymbol{\kappa}_s$ in the latent space and a linear transformation, implemented by $\mathbf{A}_s$, of $\mathbf{f}_s(\mathbf{x}_n)$. From now on we will use homogeneous coordinates and write: $\mathbf{L}_s = [\mathbf{A}_s \boldsymbol{\kappa}_s]$ and $\mathbf{z}_{ns} = [\mathbf{f}_s(\mathbf{x}_n)^\top 1]^\top$, and thus $\mathbf{g}_{ns} = \mathbf{L}_s \mathbf{z}_{ns}$. Consider the posterior distribution on latent coordinates given some data:

$$p(\mathbf{g}|\mathbf{x}) = \sum_s p(s, \mathbf{g}|\mathbf{x}) = \sum_s p(s|\mathbf{x}) p(\mathbf{g}|\mathbf{x}, s). \tag{2}$$

Given a fixed set of local feature extractors and a corresponding activities, we are interested in finding linear maps $\mathbf{L}_s$ that give rise to 'consistent' projections of the data in the latent space. By 'consistent', we mean that the $p(\mathbf{g}|\mathbf{x}, s)$ are similar for components with large posterior. If the predictions are in perfect agreement for a point $\mathbf{x}_n$, then all the $\mathbf{g}_{ns}$ are equal and the posterior $p(\mathbf{g}|\mathbf{x})$ is Gaussian, in general $p(\mathbf{g}|\mathbf{x})$ is a mixture of Gaussians. To measure the consistency, we define the following error function:

$$\Phi(\{\mathbf{L}_1, \ldots, \mathbf{L}_k\}) = \min_{\{Q_n, \ldots Q_N\}} \sum_{n,s} q_{ns} \mathcal{D}(Q_n(\mathbf{g}) \parallel p(\mathbf{g}|\mathbf{x}_n, s)), \tag{3}$$

where we used $q_{ns}$ as a shorthand for $p(s|\mathbf{x}_n)$ and $Q_n$ is a Gaussian with mean $\mathbf{g}_n$ and covariance matrix $\boldsymbol{\Sigma}_n$. The objective sums for each data point $\mathbf{x}_n$ and model $s$ the Kullback-Leibler divergence $\mathcal{D}$ between a single Gaussian $Q_n(\mathbf{g})$ and the component densities $p(\mathbf{g}|\mathbf{x}, s)$, weighted by the posterior $p(s|\mathbf{x}_n)$. It is easy to derive that in order to minimize the objective $\Phi$ w.r.t. $\mathbf{g}_n$ and $\boldsymbol{\Sigma}_n$ we obtain:

$$\mathbf{g}_n = \sum_s q_{ns} \mathbf{g}_{ns} \quad \text{and} \quad \boldsymbol{\Sigma}_n = \sigma^2 \mathbf{I}, \tag{4}$$

where $\mathbf{I}$ denotes the identity matrix. Skipping some additive and multiplicative constants with respect to the linear maps $\mathbf{L}_s$, the objective $\Phi$ then simplifies to:

$$\Phi = \sum_{n,s} q_{ns} \parallel \mathbf{g}_n - \mathbf{g}_{ns} \parallel^2 = \frac{1}{2} \sum_{n,s,t} q_{ns} q_{nt} \parallel \mathbf{g}_{nt} - \mathbf{g}_{ns} \parallel^2 \geq 0. \tag{5}$$

The main attraction with this setup is that our objective is a quadratic function of the linear maps $\mathbf{L}_s$, as in [7, 8]. Using some extra notation, we obtain a clearer form of the objective as a function of the linear maps. Let:

$$\mathbf{u}_n = [q_{n1}\mathbf{z}_{n1}^\top \ldots q_{nk}\mathbf{z}_{nk}^\top], \qquad \mathbf{U} = [\mathbf{u}_1^\top \ldots \mathbf{u}_N^\top]^\top, \qquad \mathbf{L} = [\mathbf{L}_1 \ldots \mathbf{L}_k]^\top. \tag{6}$$

Note that from (4) and (6) we have: $\mathbf{g}_n = (\mathbf{u}_n \mathbf{L})^\top$. The expected projection coordinates can thus be computed as: $\mathbf{G} = [\mathbf{g}_1 \ldots \mathbf{g}_N]^\top = \mathbf{U}\mathbf{L}$. We define the block-diagonal matrix $\mathbf{D}$ with $k$ blocks given by $\mathbf{D}_s = \sum_n q_{ns}\mathbf{z}_{ns}\mathbf{z}_{ns}^\top$. The objective can now be written as:

$$\Phi = \mathrm{Tr}\{\mathbf{L}^\top (\mathbf{D} - \mathbf{U}^\top \mathbf{U})\mathbf{L}\}. \tag{7}$$

The objective function is invariant to translation and rotation of the global latent space and re-scaling the latent space changes the objective monotonically, c.f. (5). To make solutions unique with respect to translation, rotation and scaling, we impose two constraints:

$$(transl.) : \bar{\mathbf{g}} = \sum_n \mathbf{g}_n/N = 0, \quad (rot. + scale) : \boldsymbol{\Sigma}_\mathbf{g} = \sum_n (\mathbf{g}_n - \bar{\mathbf{g}})(\mathbf{g}_n - \bar{\mathbf{g}})^\top/N = \mathbf{I}.$$

The columns of $\mathbf{L}$ minimizing $\Phi$ are characterized as the generalized eigenvectors:

$$(\mathbf{D} - \mathbf{U}^\top \mathbf{U})\mathbf{v} = \lambda \mathbf{U}^\top \mathbf{U}\mathbf{v} \quad \Leftrightarrow \quad \mathbf{D}\mathbf{v} = (\lambda + 1)\mathbf{U}^\top \mathbf{U}\mathbf{v}. \tag{8}$$

The value of the objective function is given by the sum of the corresponding eigenvalues $\lambda$. The smallest eigenvalue is always zero, corresponding to mapping all data into the same

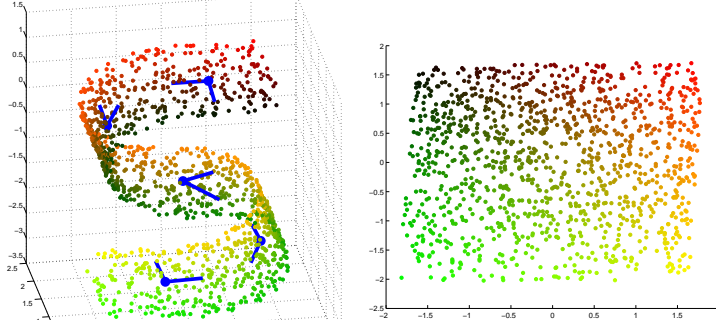

Figure 1: Data in $\mathbb{R}^3$ with local charts indicated by the axes (left). Data representation in $\mathbb{R}^2$ generated by optimizing our objective function. Expected latent coordinates $\mathbf{g}_n$ are plotted (right).

latent coordinate. This embedding is uninformative since it is constant, therefore we select the eigenvectors corresponding to the second up to the $(d+1)^{st}$ smallest eigenvalues to obtain the best embedding in $d$ dimensions. Note that, as mentioned in [7], this framework enables us to use feature extractors that provide different numbers of features.

In Fig. 1 we give an illustration of applying the above procedure to a simple manifold. The plots show the original data presented to the algorithm (left) and the 2-dimensional latent coordinates $\mathbf{g}_n = \sum_s q_{ns}\mathbf{g}_{ns}$ found by the algorithm (right).

## 3 Laplacian Eigenmaps as a special case

Consider the special case of the algorithm of Section 2, where *no* features are extracted. The only information the mixture model provides are the posterior probabilities collected in the matrix $\mathbf{Q}$ with $[\mathbf{Q}]_{ns} = q_{ns} = p(s|\mathbf{x}_n)$. In that case:

$$\mathbf{g}_{ns} = \boldsymbol{\kappa}_s, \qquad \mathbf{U} = \mathbf{Q}, \qquad \mathbf{L} = [\boldsymbol{\kappa}_1^\top \dots \boldsymbol{\kappa}_k^\top]^\top, \tag{9}$$

$$\Phi = \text{Tr}\{\mathbf{L}^\top(\mathbf{D} - \mathbf{A})\mathbf{L}\} = \sum_{s,t} \parallel \boldsymbol{\kappa}_s - \boldsymbol{\kappa}_t \parallel^2 \sum_n q_{ns}q_{nt}, \tag{10}$$

where $\mathbf{A} = \mathbf{Q}^\top\mathbf{Q}$ is an adjacency matrix with $[\mathbf{A}]_{st} = \sum_n q_{ns}q_{nt}$ and $\mathbf{D}$ is the diagonal degree matrix of $\mathbf{A}$ with $[\mathbf{D}]_{ss} = \sum_t \mathbf{A}_{st} = \sum_n q_{ns}$. Optimization under the constrains of zero mean and identity covariance leads to the generalized eigenproblem:

$$(\mathbf{D} - \mathbf{A})\mathbf{v} = \lambda \mathbf{A}\mathbf{v} \quad \Leftrightarrow \quad (\mathbf{D} - \mathbf{A})\mathbf{v} = \frac{\lambda}{1+\lambda}\mathbf{D}\mathbf{v} \tag{11}$$

The optimization problem is exactly the Laplacian Eigenmaps algorithm[4], but applied on the mixture components instead of the data points. Since we do not use any feature extractors in this setting, it can be applied to mixture models that model data for which it is hard to design feature extractors, e.g. data that has (both numerical and) categorical features. Thus, we can use mixture densities without latent variables, e.g. mixtures of multinomials, mixtures of Hidden Markov Models, etc. Notice that in this manner the mixture model not only provides a soft grouping of the data through the posteriors, but also an adjacency matrix between the groups.

## 4 Non-linear CCA by aligning local feature extractors

Canonical Correlation Analysis (CCA) is a data analysis method that finds correspondences between two or more sets of measurements. The data are provided in tuples of corresponding measurements in the different spaces. The sets of measurements can be obtained by

employing different sensors to make measurements of some phenomenon. Our main interest in this paper is to develop a nonlinear extension of CCA which works when the different measurements come from separate nonlinear manifolds that share an underlying global coordinate system. Non-linear CCA can be trained to find a shared low dimensional embedding for both manifolds, exploiting the pairwise correspondence provided by the data set. Such models can then be used for different purposes, like sensor fusion, denoising, filling in missing data, or predicting a measurement in one space given a measurement in the other space. Another important aspect of this learning setup is that the use of multiple sensors might also function as regularization helping to avoid overfitting, c.f. [11].

In CCA two (zero mean) sets of points are given: $\mathbf{X} = \{\mathbf{x}_1, \ldots, \mathbf{x}_N\} \subset \mathbb{R}^p$ and $\mathbf{Y} = \{\mathbf{y_1}, \ldots, \mathbf{y_N}\} \subset \mathbb{R}^q$. The aim is to find linear maps $\mathbf{a}$ and $\mathbf{b}$, that map members of $\mathbf{X}$ and $\mathbf{Y}$ respectively on the real line, such that the correlation between the linearly transformed variables is maximized. This is easily shown to be equivalent to minimizing:

$$\mathcal{E} = \frac{1}{2} \sum_n [\mathbf{a}\mathbf{x}_n - \mathbf{b}\mathbf{y}_n]^2 \qquad (12)$$

under the constraint that $\mathbf{a}[\sum_n \mathbf{x}_n \mathbf{x}_n^\top]\mathbf{a}^\top + \mathbf{b}[\sum_n \mathbf{y}_n \mathbf{y}_n^\top]\mathbf{b}^\top = 1$. The above is easily generalized such that the sets do not need to be zero mean and allowing a translation as well. We can also generalize by mapping to $\mathbb{R}^d$ instead of the real line, and then requiring the sum of the covariance matrices of the projections to be identity. CCA can also be readily extended to take into account more than two point sets, as we now show.

In the generalized CCA setting with multiple point-sets, allowing translations and linear mappings to $\mathbb{R}^d$, the objective is to minimize the squared distance between all pairs of projections under the same constraint as above. We denote the projection of the $n$-th point in the $s$-th point-set as $\mathbf{g}_{ns}$ and let $\mathbf{g}_n = \frac{1}{k} \sum_s \mathbf{g}_{ns}$. We then minimize the error function:

$$\Phi_{CCA} = \frac{1}{2k^2} \sum_{n,s,t} \| \mathbf{g}_{ns} - \mathbf{g}_{nt} \|^2 = \frac{1}{k} \sum_{n,s} \| \mathbf{g}_{ns} - \mathbf{g}_n \|^2 . \qquad (13)$$

The objective $\Phi$ in equation (5) coincides with $\Phi_{CCA}$ if $q_{ns} = 1/k$. The different constraints imposed upon the optimization by CCA and our objective of the previous sections are equivalent. We can thus regard the alignment procedure as a *weighted* form of CCA. This suggests using the coordination technique for non-linear CCA. This is achieved quite easily, without modifying the objective function (5). We consider different point sets, each having a mixture of locally valid linear projections into the 'global' latent space that is now shared by all mixture components and point sets. We minimize the weighted sum of the squared distances between *all pairs* of projections, i.e. we have pairs of projections due to the same point set and also pairs that combine projections from different point sets.

We use $c$ as an index ranging over the $C$ different observation spaces, and write $q_{ns}^c$ for the posterior on component $s$ for observation $n$ in observation space $c$. Similarly, we use $\mathbf{g}_{ns}^c$ to denote the projection due component $s$ from space $c$. The average projection due to observation space $c$ is then denoted by $\mathbf{g}_n^c = \sum_s q_{ns}^c \mathbf{g}_{ns}^c$. We use index $r$ to range over all mixture components and observation spaces, so that $q_{nr} = \frac{1}{C} p(s|\mathbf{x}_n)$ if $r$ corresponds to $(c = 1, s)$ and $q_{nr} = \frac{1}{C} p(s|\mathbf{y}_n)$ if $r$ corresponds to $(c = 2, s)$, i.e. $r \leftrightarrow (c, s)$. The overall average projection then becomes: $\mathbf{g}_n = \frac{1}{C} \sum_c \mathbf{g}_n^c = \sum_r q_{nr}\mathbf{g}_{nr}$. The objective (5) can now be rewritten as:

$$\Phi = \sum_{n,r} q_{nr} \| \mathbf{g}_{nr} - \mathbf{g}_n \|^2 = \frac{1}{C} \sum_{c,n} \| \mathbf{g}_n - \mathbf{g}_n^c \|^2 + \frac{1}{C} \sum_{c,n,s} q_{ns}^c \| \mathbf{g}_n^c - \mathbf{g}_{ns}^c \|^2 . \quad (14)$$

Observe how in (14) the objective sums *between* point set consistency of the projections (first summand) and *within* point set consistency of the projections (second summand).

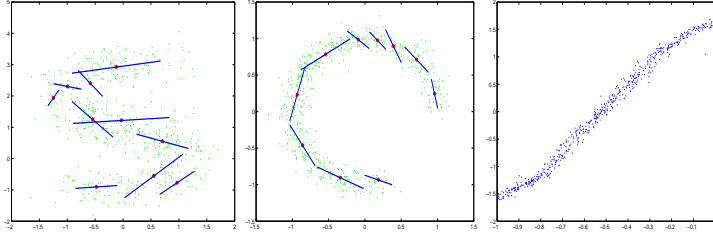

Figure 2: Data and charts, indicated by bars (left-middle). Latent coordinates (vert.) and coordinate on generating curve (hor.) (right).

The above technique can also be used to get more stable results of the chart coordination procedure for a single manifold discussed in Section 2. Robustness for variation in the mixture fitting can be improved by using several sets of charts fitted to the same manifold. We can then align all these sets of charts by optimizing (14). This aligns the charts within each set and at the same time makes sure the different sets of aligned charts are aligned, providing important regularization, since now every point is modeled by several local models.

Note that if the charts and responsibilities are obtained using a mixture of PCA or factor analyzers, the local linear mappings to the latent space induce a Gaussian mixture in the latent space. This mixture can be used to compute responsibilities on components given latent coordinates. Also, for each linear map from the data to the latent space we can compute a pseudo inverse projecting back. By averaging the individual back projections with the responsibilities computed in latent space we obtain a projection from the latent space to the data space. In total, we can thus map from one observation space into another. This is how we generated the reconstructions in the experiments reported below. When using linear CCA for data that is non-linearly embedded, reconstructions will be poor since linear CCA can only map into a low dimensional linear subspace.

As an illustrative example of the non-linear CCA we used two point-sets in $\mathbb{R}^2$. The first point-set was generated on an S-shaped curve the second point set was generated along an arc, see Fig. 2. To both point sets we added Gaussian noise and we learned a 10 component mixture model on both sets. In the rightmost panel of Fig. 2 the, clearly successfully, discovered latent coordinates are plotted against the coordinate on the generating curve. Below, we describe three more challenging experiments.

In the first experiment we use two data sets which we know to share the same underlying degrees of freedom. We use images of a face varying its gaze left-right and up-down. We cut these images in half to obtain our two sets of images. We trained the system on 1500 image halves of $40 \times 20$ pixels each. Both image halves were modeled with a mixture of 40 components. In Fig. 3 some generated right half images based on the left half are shown.

The second experiment concerns appearance based pose estimation of an object. One point set consists of a pixel representation of images of an object and the other point set contains the corresponding pose of the camera w.r.t. the object. For the pose parameters we used the identity to 'extract' features (i.e. we just used one component for this space). The training data was collected[1] by moving a camera over the half-sphere centered at the object. A mixture of 40 PCA's was trained on the image data and aligned with the pose parameters in a 2-dimensional latent space. The right panel of Fig. 3 shows reconstructions of the images conditioned on various pose inputs (left image of each pair is reconstruction based on pose of right image). Going the other way, when we input an image and estimate the pose, the absolute errors in the longitude $(0° - 360°)$ were under $10°$ in over 80% of the cases and for latitude $(0° - 90°)$ this was under $5°$ in over 90% of the cases.

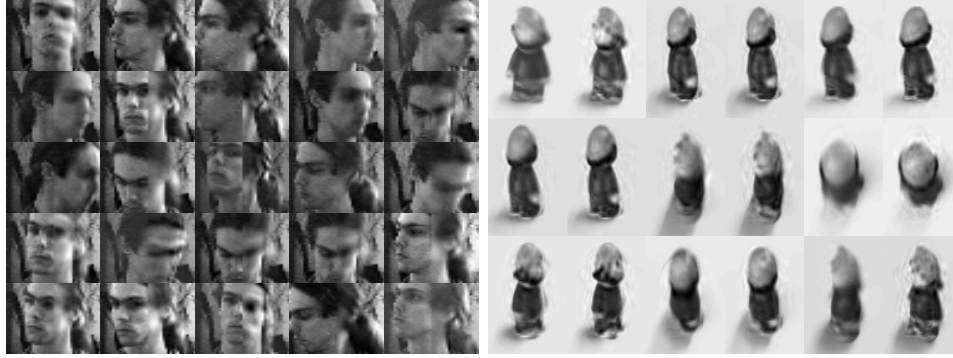

Figure 3: Right half of the images was generated given the left half using the trained model (left). Image reconstructions given pose parameters (right).

In the third experiment we use the same images as in the second experiment, but replace the direct (low dimensional) supervision signal of the pose parameters with (high dimensional) correspondences in the form of images of another object in corresponding poses. We trained a mixture of 40 PCA's on both image sets (2000 images of $64 \times 64$ pixels in each set) and aligned these in a 3-dimensional latent space. Comparing the pose of an object to the pose of the nearest (in latent space) image from the other object the std. dev. of error in latitude is $2.0°$. For longitude we found 4 errors of about $180°$ in our 500 test cases, the rest of the errors had std. dev. $3.9°$. Given a view of one object we can reconstruct the corresponding view of the second object, Fig. 4 shows some of the obtained reconstruction results. All presented reconstructions were made for data not included in training.

## 5  Discussion

In this paper, we have extended alignment methods for single manifold nonlinear dimensionality reduction to perform non-linear CCA using measurements from multiple manifolds. We have also shown the close relationship with Laplacian Eigenmaps[4] in the degenerate case of a single manifold and feature extractors of zero dimensionality.

In [7] a related method to coordinate local charts is proposed, which is based on the LLE cost function as opposed to our cross-entropy term; this means that we need more than just a set of local feature extractors and their posteriors: we also need to be able to compute reconstruction weights, collected in a $N \times N$ weight matrix. The weights indicate how we can reconstruct each data point from its nearest neighbors. Computing these weights requires access to the original data directly, not just through the "interface" of the mixture model. Defining sensible weights and the 'right' number of neighbors might not be straightforward, especially for data in non-Euclidean spaces. Furthermore, computing the weights costs in principle $O(N^2)$ because we need to find nearest neighbors, whereas the presented work has running time linear in the number of data points.

In [11] it is considered how to find low dimensional representations for multiple point sets simultaneously, given few correspondences between the point sets. The generalization of LLE presented there for this problem is closely related to our non-linear CCA model. The work presented here can also be extended to the case where we know only for few points in one set to which points they correspond in the other set. The use of multiple sets of charts for one data set is similar in spirit as the self-correspondence technique of [11] where the data is split into several overlapping sets used to stabilize the generalized LLE.

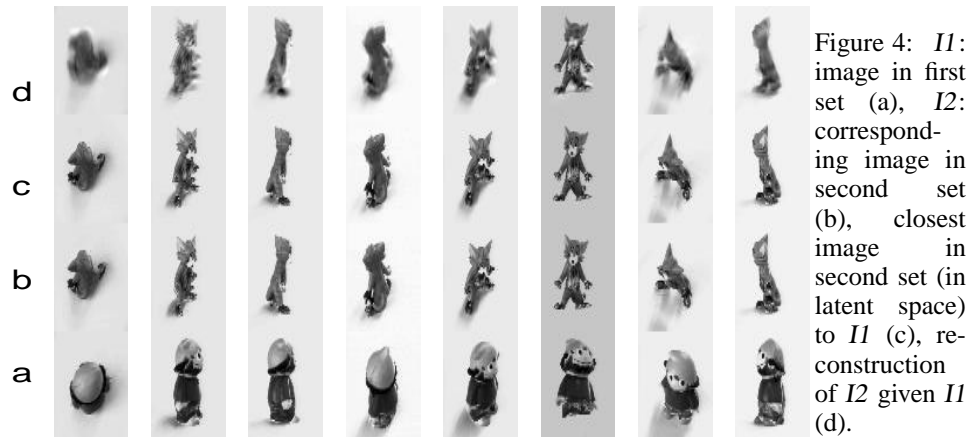

Figure 4: *I1*: image in first set (a), *I2*: corresponding image in second set (b), closest image in second set (in latent space) to *I1* (c), reconstruction of *I2* given *I1* (d).

Finally, it would be interesting to compare our approach with treating the data in the joint $(\mathbf{x}, \mathbf{y})$ space and employing techniques for a single point set[8, 7, 6]. In this case, points for which we do not have the correspondence can be treated as data with missing values.

### Acknowledgments

JJV and NV are supported by the Technology Foundation STW (AIF4997) applied science division of NWO and the technology program of the Dutch Ministry of Economic Affairs. STR is supported in part by the Learning Project of IRIS Canada and by the NSERC.

## Footnotes

[1]Thanks to G. Peters for sharing the images used in [12] and recorded at the Institute for Neural Computation, Ruhr-University Bochum, Germany.

## References

[1] J.B. Tenenbaum, V. de Silva, and J.C. Langford. A global geometric framework for nonlinear dimensionality reduction. *Science*, 290(5500):2319–2323, December 2000.

[2] S.T. Roweis and L.K. Saul. Nonlinear dimensionality reduction by locally linear embedding. *Science*, 290(5500):2323–2326, December 2000.

[3] B. Schölkopf, A.J. Smola, and K. Müller. Nonlinear component analysis as a kernel eigenvalue problem. *Neural Computation*, 10:1299–1319, 1998.

[4] M. Belkin and P. Niyogi. Laplacian eigenmaps and spectral techniques for embedding and clustering. In *Advances in Neural Information Processing Systems*, volume 14, 2002.

[5] C. Bregler and S.M. Omohundro. Surface learning with applications to lipreading. In *Advances in Neural Information Processing Systems*, volume 6, 1994.

[6] S.T. Roweis, L.K. Saul, and G.E. Hinton. Global coordination of local linear models. In *Advances in Neural Information Processing Systems*, volume 14, 2002.

[7] Y.W. Teh and S.T. Roweis. Automatic alignment of local representations. In *Advances in Neural Information Processing Systems*, volume 15, 2003.

[8] M. Brand. Charting a manifold. In *Advances in Neural Information Processing Systems*, volume 15, 2003.

[9] C.M. Bishop, M. Svensén, and C.K.I Williams. GTM: the generative topographic mapping. *Neural Computation*, 10:215–234, 1998.

[10] T. Kohonen. *Self-organizing maps.* Springer, 2001.

[11] J.H. Ham, D.D. Lee, and L.K. Saul. Learning high dimensional correspondences from low dimensional manifolds. In *ICML'03, workshop on the continuum from labeled to unlabeled data in machine learning and data mining*, 2003.

[12] G. Peters, B. Zitova, and C. von der Malsburg. How to measure the pose robustness of object views. *Image and Vision Computing*, 20(4):249–256, 2002.
